# Local Rules for Global MAP: When Do They Work ?

**Kyomin Jung**[*]
KAIST
Daejeon, Korea
kyomin@kaist.edu

**Pushmeet Kohli**
Microsoft Research
Cambridge, UK
pkohli@microsoft.com

**Devavrat Shah**
MIT
Cambridge, MA, USA
devavrat@mit.edu

## Abstract

We consider the question of computing Maximum A Posteriori (MAP) assignment in an arbitrary pair-wise Markov Random Field (MRF). We present a randomized iterative algorithm based on simple local updates. The algorithm, starting with an arbitrary initial assignment, updates it in each iteration by first, picking a random node, then selecting an (appropriately chosen) random local neighborhood and optimizing over this local neighborhood. Somewhat surprisingly, we show that this algorithm finds a near optimal assignment within $n \log^2 n$ iterations with high probability for *any* $n$ node pair-wise MRF with *geometry* (i.e. MRF graph with polynomial growth) with the approximation error depending on (in a reasonable manner) the geometric growth rate of the graph and the average radius of the local neighborhood – this allows for a graceful tradeoff between the complexity of the algorithm and the approximation error. Through extensive simulations, we show that our algorithm finds extremely good approximate solutions for various kinds of MRFs with geometry.

## 1 Introduction

The abstraction of Markov random field (MRF) allows one to utilize graphical representation to capture inter-dependency between large number of random variables in a succinct manner. The MRF based models have been utilized successfully in the context of coding (e.g. the low density parity check code [15]), statistical physics (e.g. the Ising model [5]), natural language processing [13] and image processing in computer vision [11, 12, 19]. In most applications, the primary inference question of interest is that of finding maximum a posteriori (MAP) solution – e.g. finding a most likely transmitted message based on the received signal.

**Related Work.** Computing the exact MAP solution in general probabilistic models is an NP-hard problem. This had led researchers to resort of fast approximate algorithms. Various such algorithmic approaches have been developed over more than the past three decades. In essence, all such approaches try to find a locally optimal solution of the problem through iterative procedure. These "local update" algorithms start from an initial solution and proceed by making a series of changes which lead to solutions having lower energy (or better likelihood), and hence are also called "move making algorithms". At each step, the algorithms search the space of all possible local changes that can be made to the current solution (also called move space), and choose the one which leads to the solution having the highest probability or lowest energy.

One such algorithm (which has been rediscovered multiple times) is called Iterated Conditional Modes or ICM for short. Its local update involves selecting (randomly or deterministically) a variable of the problem. Keeping the values of all other variables fixed, the value of the selected variable

---

[*]This work was partially carried out while the author was visiting Microsoft Research Cambridge, and was partially supported by NSF CAREER project CNS-0546590.

is chosen which results in a solution with the maximum probability. This process is repeated by selecting other variables until the probability cannot be increased further.

The size of the move space is the defining characteristic of any such move making algorithm. A large move space means that more extensive changes to the current solution can be made. This makes the algorithm less prone to getting stuck in local minima and also results in a faster rate of convergence. Expansion and Swap are move making algorithms which search for the optimal move in a move space of size $2^n$ where $n$ is the number of random variables. For energy functions composed of metric pairwise potentials, the optimal move can be found in polynomial time by minimizing a submodular quadratic pseudo-boolean function [3] (or solving an equivalent minimum cost st-cut problem).

The last few years have seen a lot of interest in st-mincut based move algorithms for energy minimization. Komodakis et al. [9] recently gave an alternative interpretation of the expansion algorithm. They showed that expansion can be seen as solving the dual of a linear programming relaxation of the energy minimization problem. Researchers have also proposed a number of novel move encoding strategies for solving particular forms of energy functions. Veksler [18] proposed a move algorithm in which variables can choose any label from a range of labels. They showed that this move space allowed them to obtain better minima of energy functions with truncated convex pairwise terms. Kumar and Torr [10] have since shown that the range move algorithm achieves the same guarantees as the ones obtained by methods based on the standard linear programming relaxation.

A related popular algorithmic approach is based on max-product belief propagation (cf. [14] and [22]). In a sense, it can be viewed as an iterative algorithm that makes local updates based optimizing based on the immediate graphical structure. There is a long list of literature on understanding the conditions under which max-product belief propagation algorithm find correct solution. Specifically, in recent years a sequence of results suggest that there is an intimate relation between the max-product algorithm and a natural linear programming relaxation – for example, see [1, 2, 8, 16, 21].

We also note that Swendsen-Wang algorithm (SW) [17], a local flipping algorithm, has a philosophy similar to ours in that it repeats a process of randomly partitioning the graph, and computing an assignment. However, the graph partitioning of SW is fundamentally different from ours and there is no known guarantee for the error bound of SW.

In summary, all the approaches thus far with provable guarantees for local update based algorithm are primarily for linear or more generally convex optimization setup.

**Our Contribution.** As the main result of this paper, we propose a randomized iterative local algorithm that is based on simple local updates. The algorithm, starting with an arbitrary initial assignment, updates it in each iteration by first picking a random node, then its (appropriate) random local neighborhood and optimizing over this local neighborhood. Somewhat surprisingly, we show that this algorithm finds near optimal assignment within $n \log^2 n$ iterations with high probability for graphs with *geometry* – i.e. graphs in which the neighborhood of each node within distance $r$ grows no faster than a polynomial in $r$. Such graphs can have arbitrarily structure subject to this polynomial growth structure. We show that the approximation error depends gracefully on the average random radius of the local neighborhood and degree of polynomial growth of the graph. Overall, our algorithm can provide an $\varepsilon-$approximation MAP with $C(\varepsilon) n \log^2 n$ total computation with $C(\varepsilon)$ depending only on $\varepsilon$ and the degree of polynomial growth. The crucial novel feature of our algorithm is the appropriate selection of random local neighborhood rather than deterministic in order to achieve provable performance guarantee.

We note that near optimality of our algorithm does not depend on convexity property, or tree-like structure as many of the previous works; but only relies on geometry of the graphical structure which is present in many graphical models of interest such as those arising in image processing, wireless networks, etc.

We use our algorithm to verify its performance in simulation scenario. Specifically, we apply our algorithm to two popular setting: (a) a grid graph based pairwise MRF with varying node and edge interaction strengths, and (b) a grid graph based MRF on the weighted independent set (or hardcore) model. We find that with very small radius (within 3), we find assignment which within 1% (0.99 factor) of the MAP for a large range of parameters and upto graph of 1000 nodes.

**Organization.** We start by formally stating our problem statement and main theorem (Theorem 1) in Section 2. This is followed by a detailed description of the algorithm in Section 3. We present the sketch proof of the main result in Section 4. Finally, we provide a detailed simulation results in Section 5.

## 2 Main Results

We start with the formal problem description and useful definitions/notations followed by the statement of the main result about performance of the algorithm. The algorithm will be stated in the next section.

**Definitions & Problem Statement.** Our interest is in a pair-wise MRF defined next. We note that, formally all (non pair-wise) MRFs are equivalent to pair-wise MRFs – e.g. see [20].

**Definition 1** (Pair-wise MRF). *A pair-wise MRF based on graph $G = (V, E)$ with $n = |V|$ vertices and edge set $E$ is defined by associated a random variable $X_v$ with each vertex $v \in V$ taking value in finite alphabet set $\Sigma$; the joint distribution of $\mathbf{X} = (X_v)_{v \in V}$ defined as*

$$\Pr[\mathbf{X} = \mathbf{x}] \quad \propto \quad \prod_{v \in V} \Psi_v(x_v) \cdot \prod_{(u,v) \in E} \Psi_{uv}(x_u, x_v) \tag{1}$$

*where $\Psi_v : \Sigma \to \mathbb{R}_+$ and $\Psi_{uv} : \Sigma^2 \to \mathbb{R}_+$ are called node and edge potential functions.* [1]

In this paper, the question of interest is to find the maximum a posteriori (MAP) assignment $\mathbf{x}^* \in \Sigma^n$, i.e.

$$\mathbf{x}^* \in \arg \max_{\mathbf{x} \in \Sigma^n} \Pr[\mathbf{X} = \mathbf{x}].$$

Equivalently, from the optimization point of view, we wish to find an optimal assignment of the problem

$$\text{maximize} \quad H(\mathbf{x}) \qquad \text{over} \qquad \mathbf{x} \in \Sigma^n,$$

where

$$H(\mathbf{x}) = \sum_{v \in V} \ln \Psi_v(x_v) + \sum_{(u,v) \in E} \ln \Psi_{uv}(x_u, x_v).$$

For completeness and simplicity of exposition, we assume that the function $H$ is finite valued over $\Sigma^n$. However, results of this paper extend for hard constrained problems such as the *hardcore* or *independent set* model.

In this paper, we will design algorithms for finding approximate MAP problem. Specifically, we call an assignment $\widehat{\mathbf{x}}$ as an $\varepsilon$-approximate MAP if

$$(1 - \varepsilon) H(\mathbf{x}^*) \leq H(\widehat{\mathbf{x}}) \leq H(\mathbf{x}^*).$$

**Graphs with Geometry.** We define notion of graphs with geometry here. To this end, a graph $G = (V, E)$ induces a natural 'graph metric' on vertices $V$, denoted by $\mathbf{d_G} : V \times V \to \mathbb{R}_+$ with $\mathbf{d_G}(v, u)$ as the length of the shortest path between $u$ and $v$; with it defined as $\infty$ if there is no path between them.

**Definition 2** (Graph with Polynomial Growth). *We call a graph $G$ with polynomial growth of degree (or growth rate) $\rho$, if for any $v \in V$ and $r \in \mathbb{N}$,*

$$|\mathbf{B}_G(v, r)| \leq C \cdot r^\rho,$$

*where $C > 0$ is a universal constant and $\mathbf{B}_G(v, r) = \{w \in V | \mathbf{d}_G(w, v) < r\}$.*

A large class of graph model naturally fall into the graphs with polynomial growth. To begin with, the standard $d$-dimensional regular grid graphs have polynomial growth rate $d$ – e.g. $d = 1$ is the line graph. More generally, in recent years in the context of computational geometry and metric embedding, the graphs with finite doubling dimensions have become popular object of study [6, 7].

It can be checked that a graph with doubling dimension $\rho$ is also a graph with polynomial growth rate $\rho$. Finally, the popular geometric graph model where nodes are placed arbitrarily on a two dimensional surface with minimum distance separation and two nodes have an edge between them if they are within certain finite distance, then it is indeed a graph with finite polynomial growth rate.

**Statement of Main Result.** The main result of this paper is a randomized iterative algorithm based on simple local updates. In essence the algorithm works as follows. It starts with an arbitrary initial assignment. In each iteration, it picks a node, say $v$ from all $n$ nodes of $V$, uniformly at random and picks a random radius $Q$ (as per specific distribution). The algorithm re-assigns values to all nodes within distance $Q$ of node $v$ with respect to graph distance $\mathbf{d_G}$ by finding the optimal assignment for this local neighborhood subject to keeping the assignment to all other nodes the same. The algorithm Loc-Algo described in Section 3 repeats this process for $n \log^2 n$ many times. We show that Loc-Algo finds near optimal solution with high probability as long as the graph has finite polynomial growth rate.

**Theorem 1.** *Given MRF based on graph $G = (V, E)$ of $n = |V|$ nodes with polynomial growth rate $\rho$ and approximation parameter $\varepsilon \in (0,1)$, our algorithm* Loc-Algo *with $O\left(\log(1/\delta)n \log n\right)$ iterations produces a solution $\widehat{\mathbf{x}}$ such that*

$$\Pr[H(\mathbf{x}^*) - H(\widehat{\mathbf{x}}) \leq 2\varepsilon H(\mathbf{x}^*)] \geq 1 - \delta - \frac{1}{poly(n)}.$$

*And each iteration takes at most $\zeta(\varepsilon, \rho)$ computation, with*

$$\zeta(\varepsilon, \rho) \leq |\Sigma|^{CK(\varepsilon,\rho)^\rho},$$

*where $K(\varepsilon, \rho)$ is defined as*

$$K = K(\varepsilon, \rho) = \frac{8\rho}{\varphi} \log\left(\frac{8\rho}{\varphi}\right) + \frac{4}{\varphi} \log C + \frac{4}{\varphi} \log \frac{1}{\varphi} + 2 \quad with \quad \varphi = \frac{\varepsilon}{5C2^\rho}.$$

In a nutshell, Theorem 1 say that the complexity of the algorithm for obtaining an $\varepsilon$-approximation scales almost linearly in $n$, double exponentially in $1/\varepsilon$ and $\rho$. On one hand, this result establishes that it is indeed possible to have polynomial (or almost linear) time approximation algorithm for arbitrary pair-wise MRF with polynomial growth. On the other hand, though theoretical bound on the pre-constant $\zeta(\varepsilon, \rho)$ as function of $1/\varepsilon$ and $\rho$ is not very exciting, our simulations suggest (see Section 5) that even for hard problem setup, the performance is much more optimistic than predicted by these theoretical bounds. Therefore, as a recommendation for a system designer, we suggest use of smaller 'radius' distribution in algorithm described in Section 3 for obtaining good algorithm.

## 3 Algorithm Description

In this section, we provide details of the algorithm intuitively described in the previous section. As noted earlier, the algorithm iteratively updates its estimation of MAP, denoted by $\widehat{\mathbf{x}}$. Initially, the $\widehat{\mathbf{x}}$ is chosen arbitrarily. Iteratively, at each step a vertex $v \in V$ is chosen uniformly at random along with a random radius $Q$ that is chosen independently as per distribution $\mathbf{Q}$. Then, select $\mathcal{R} \subset V$, the local neighborhood (or ball) of radius $Q$ around $v$ as per graph distance $\mathbf{d_G}$, i.e. $\{w \in V | \mathbf{d_G}(u, w) < Q\}$. Then while keeping the assignment of all nodes in $V \backslash \mathcal{R}$ fixed as per $\widehat{\mathbf{x}} = (\hat{x}_v)_{v \in V}$, find MAP assignment $\mathbf{x}^{*,\mathcal{R}}$ restricted to nodes of $\mathcal{R}$. And, update the assignment of nodes in $v \in \mathcal{R}$ as per $\mathbf{x}^{*,\mathcal{R}}$. A caricature of an iteration is described in Figure 1. The precise description of the algorithm is given in Figure 2.

In order to have good performance, it is essential to choose appropriate distribution $\mathbf{Q}$ for selection of random radius $Q$ each time. Next, we define this distribution which is essentially a *truncated Geometric* distribution. Specifically, given parameters $\varepsilon \in (0,1)$ and the polynomial growth rate $\rho$ (with constant $C$) of the graph, define $\varphi = \frac{\varepsilon}{5C2^\rho}$, and

$$K = K(\varepsilon, \rho) = \frac{8\rho}{\varphi} \log\left(\frac{8\rho}{\varphi}\right) + \frac{4}{\varphi} \log C + \frac{4}{\varphi} \log \frac{1}{\varphi} + 2.$$

Then, the distribution (or random variable) $\mathbf{Q}$ is defined over integers from $1$ to $K(\varepsilon, \rho)$ as

$$\Pr[\mathbf{Q} = i] = \begin{cases} \varphi(1-\varphi)^{i-1} & \text{if } 1 \leq i < K(\varepsilon, \rho) \\ (1-\varphi)^{K-1} & \text{if } i = K(\varepsilon, \rho) \end{cases}.$$

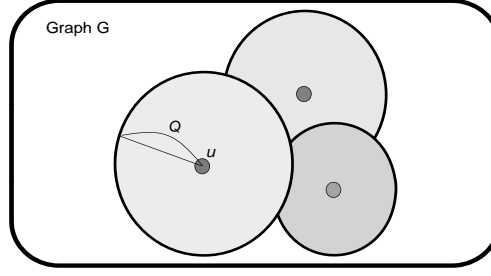

$$\Pr[Q = i] = \varepsilon(1-\varepsilon)^{i-1} \quad \text{for } i = 1,2,3\ldots,(K-1)$$

Figure 1: Pictorial description of an iteration of LOC-ALGO.

LOC-ALGO($\varepsilon, K$)

> (0) Input: MRF $G = (V, E)$ with $\phi_i(\cdot), i \in V, \psi_{ij}(\cdot, \cdot), (i,j) \in E$.
>
> (1) Initially, select $\widehat{\mathbf{x}} \in \Sigma^n$ arbitrarily.
>
> (2) Do the following for $n \log^2 n$ many times :
>
>> (a) Choose an element $u \in V$ uniformly at random.
>> (b) Draw a random number $Q$ according to the distribution $\mathbf{Q}$.
>> (c) Let $\mathcal{R} \leftarrow \{w \in V | \mathbf{d_G}(u, w) < Q\}$.
>> (d) Through dynamic programming (or exhaustive computation) find an exact MAP $\mathbf{x}^{*,\mathcal{R}}$ for $\mathcal{R}$ while fixing all the other assignment of $\widehat{\mathbf{x}}$ value outside $\mathcal{R}$.
>> (e) Change values of $\widehat{\mathbf{x}}$ for $\mathcal{R}$ by $\mathbf{x}^{*,\mathcal{R}}$.
>
> (3) Output $\widehat{\mathbf{x}}$.

Figure 2: Algorithm for approximate MAP computation.

## 4 Proof of Theorem 1

In this section, we present proof of Theorem 1. To that end, we will prove the following Lemma.

**Lemma 1.** *If we run the* LOC-ALGO *with* $(2n \ln n)$ *iterations, with probability at least* $1 - 1/n$, *we have*

$$(1 - \varepsilon)H(\mathbf{x}^*) \le \mathbb{E}[H(\widehat{\mathbf{x}})] \le H(\mathbf{x}^*).$$

From Lemma 1, we obtain Theorem 1 as follows. Define $T = 2\log(1/\delta)$, and consider LOC-ALGO with $(2Tn \ln n)$ iterations. From the fact that $H(\mathbf{x}^*) - H(\widehat{\mathbf{x}}) \ge 0$, and by the Markov inequality applied to $H(\mathbf{x}^*) - H(\widehat{\mathbf{x}})$ with Lemma 1, we have that after $(2n \ln n)$ iterations,

$$\Pr[H(\mathbf{x}^*) - H(\widehat{\mathbf{x}}) \le 2\varepsilon H(\mathbf{x}^*)] \ge \frac{1}{2}. \tag{2}$$

Note that (2) is true for any initial assignment of LOC-ALGO. Hence for each $1 \le t \le T$, after $(2tn \ln n)$ iterations, (2) holds independently with probability $1 - 1/n$. Also, note that $H(\widehat{\mathbf{x}})$ is increasing monotonically. Hence, $H(\mathbf{x}^*) - H(\widehat{\mathbf{x}}) > 2\varepsilon H(\mathbf{x}^*)$ holds after $(2Tn \ln n)$ iterations only if the same holds after $(2tn \ln n)$ iterations for all $1 \le t \le T$. Hence, after $(2Tn \ln n)$ iterations, we have $\Pr[H(\mathbf{x}^*) - H(\widehat{\mathbf{x}}) \le 2\varepsilon H(\mathbf{x}^*)] \ge 1 - \delta - 1/\text{poly}(n)$, which proves the first part of Theorem 1.

For the total computation bound in Theorem 1, note that each iteration of LOC-ALGO involves dynamic programming over a local neighborhood of radius at most $K = K(\varepsilon, \rho)$ around a node.

This involves, due to the polynomial growth condition, at most $CK^\rho$ nodes. Each variable can takes at most $|\Sigma|$ different values. Therefore, dynamic programming (or exhaustive search) can take at most $|\Sigma|^{CK^\rho}$ operations as claimed.

**Proof of Lemma 1.** First observe that by the standard argument in the classical coupon collector problem with $n$ coupons (e.g. see [4]), it follows that after $2n \ln n$ iterations, with probability at least $1 - 1/n$, all the vertices of $V$ will be chosen as 'ball centers' at least once.

**Error bound.** Now we prove that if all the vertices of $V$ are chosen as 'ball centers' at least once, the answer $\widehat{\mathbf{x}}$ generated by LOC-ALGO after $2n \ln n$ iterations, is indeed an $\varepsilon$-approximation on average. To this end, we construct an imaginarily set of edges as follows. Imagine that the procedure (2) of LOC-ALGO is done with an iteration parameter $t \in \mathbb{Z}_+$. Then for each vertex $v \in V$, we assign the largest iteration number $t$ such that the chosen ball $\mathcal{R}$ at the iteration $t$ contains $w$. That is,

$$T(v) = \max\{t \in \mathbb{Z}_+ | \text{LOC-ALGO chooses } v \text{ as a member of } \mathcal{R} \text{ at iteration } t\}.$$

Clearly, this is well defined algorithm is run till each node is chosen as the 'ball center' at least once. Now define an *imaginary boundary set* of LOC-ALGO as

$$\mathcal{B} = \{(u, w) \in E | T(u) \neq T(w)\}.$$

Now consider graph $G' = (V, E \backslash \mathcal{B})$ obtained by removing edges $\mathcal{B}$ from $G$. In this graph, nodes of the same connected component have same $T(\cdot)$ value. Next, we state two Lemmas that will be crucial to the proof of the Theorem. Proof of Lemmas 2 and 3 are omitted.

**Lemma 2.** *Given two MRFs $\mathbf{X}_1$ and $\mathbf{X}_2$ on the same graph $\mathcal{G} = (\mathcal{V}, \mathcal{E})$ with identical edge potentials $\{\psi_{ij}(\cdot, \cdot)\}$, $(i, j) \in \mathcal{E}$ but distinct node potentials $\{\phi_i^1(\cdot)\}, \{\phi_i^2(\cdot)\}, i \in \mathcal{V}$ respectively. For each $i \in \mathcal{V}$, define $\phi_i^D = \max_{\sigma \in \Sigma} |\phi_i^1(\sigma) - \phi_i^2(\sigma)|$. Finally, for $\ell \in \{1, 2\}$ and any $\mathbf{x} \in \Sigma^n$, define $H_\ell(\mathbf{x}) = \sum_{i \in \mathcal{V}} \phi_i^\ell(x_i) + \sum_{(i,j) \in \mathcal{E}} \psi_{ij}(x_i, x_j)$, with $\mathbf{x}^{*,\ell}$ being a MAP assignment of MRF $\mathbf{x}_\ell$. Then, we have $|H_1(\mathbf{x}^{*,1}) - H_1(\mathbf{x}^{2,*})| \leq 2 \left( \sum_{i \in \mathcal{V}} \phi_i^D \right)$.*

**Lemma 3.** *Given MRF $\mathbf{X}$ defined on $G$ (as in (1)), the algorithm LOC-ALGO produces output $\widehat{\mathbf{x}}$ such that*

$$|H(\mathbf{x}^*) - H(\widehat{\mathbf{x}})| \leq 5 \left( \sum_{(i,j) \in \mathcal{B}} \left( \psi_{ij}^U - \psi_{ij}^L \right) \right),$$

*where $\mathcal{B}$ is the (random) imaginary boundary set of LOC-ALGO, $\psi_{ij}^U \triangleq \max_{\sigma, \sigma' \in \Sigma} \psi_{ij}(\sigma, \sigma')$, and $\psi_{ij}^L \triangleq \min_{\sigma, \sigma' \in \Sigma} \psi_{ij}(\sigma, \sigma')$.*

Now we obtain the following lemma that utilizes the fact that the distribution $\mathbf{Q}$ follows a geometric distribution with rate $(1 - \varphi)$ – its proof is omitted.

**Lemma 4.** *For any edge $e \in E$ of $G$,*

$$\Pr[e \in \mathcal{B}] \leq \varphi.$$

From Lemma 4, we obtain that

$$\sum_{(i,j) \in \mathcal{B}} \left( \psi_{ij}^U - \psi_{ij}^L \right) \leq \varphi \sum_{(i,j) \in E} \left( \psi_{ij}^U - \psi_{ij}^L \right). \tag{3}$$

Finally, we establish the following lemma that bounds $\sum_{(i,j) \in E} \left( \psi_{ij}^U - \psi_{ij}^L \right)$ – its proof is omitted.

**Lemma 5.** *If $G$ has maximum vertex degree $d^*$, then*

$$\sum_{(i,j) \in E} \left( \psi_{ij}^U - \psi_{ij}^L \right) \leq (d^* + 1) H(\mathbf{x}^*). \tag{4}$$

Now recall that the maximum vertex degree $d^*$ of $G$ is less than $2^\rho C$ by the definition of polynomially growing graph. Therefore, by Lemma 3, (3), and Lemma 5, the output produced by the LOC-ALGO algorithm is such that

$$|H(\mathbf{x}^*) - H(\widehat{\mathbf{x}})| \leq 5(d^* + 1)\varphi H(\mathbf{x}^*) \leq \varepsilon H(\mathbf{x}^*),$$

where recall that $\varphi = \frac{\varepsilon}{5C2^\rho}$. This completes the proof of Lemma 1.

## 5  Experiments

Our algorithm provides a provable approximation for any MRF on a polynomially growing graph. In this section, we present experimental evaluations of our algorithm for two popular models: (a) synthetic Ising model, and (b) hardcore (independent set) model. As a reader will notice, the experimental results not only conform the qualitatively behavior proved by our theoretical result, but it also suggest that much tighter approximation guarantees should be expected in practice compared to what is guaranteed by theoretical results.

**Setup 1**[2] Consider a binary (i.e. $\Sigma = \{0,1\}$) MRF on an $n_1 \times n_2$ grid $G = (V, E)$:

$$\Pr(\mathbf{x}) \propto \exp\left(\sum_{i \in V} \theta_i x_i + \sum_{(i,j) \in E} \theta_{ij} x_i x_j\right), \quad \text{for } \mathbf{x} \in \{0,1\}^{n_1 n_2}.$$

We consider the following scenario for choosing parameters (with the notation $\mathcal{U}[a,b]$ for the uniform distribution over the interval $[a,b]$):

1. For each $i \in V$, choose $\theta_i$ independently as per the distribution $\mathcal{U}[-1, 1]$.
2. For each $(i, j) \in E$, choose $\theta_{ij}$ independently from $\mathcal{U}[-\alpha, \alpha]$. Here the *interaction* parameter $\alpha$ is chose from $\{0.125, 0.25, 0.5, 1, 2, 4, 8, 16, 32, 64\}$.

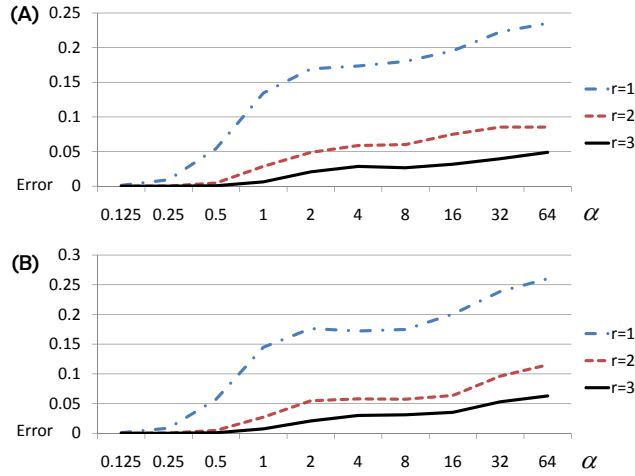

Figure 3: (A) plots the error of local update algorithm for a random Ising model in the grid graph of size $10 \times 10$, and (B) plots the error in the grid of size $100 \times 10$.

To compare the effectiveness of our algorithm for each size of the local updates, in our simulation, we fix the square size as a constant instead of choosing it from a distribution. We run the simulation for the local square size $r \times r$ with $r = 1, 2, 3$, where $r = 1$ is the case when each square consists of a single vertex. We computed an exact MAP assignment $\mathbf{x}^*$ by dynamic programming, and computed the output $\widehat{\mathbf{x}}$ of our local update algorithm for each $r$, by doing $4n_1 n_2 \log(n_1 n_2)$ many local updates for $n_1 \times n_2$ grid graph. Then compare the error as follows:

$$\text{Error} = \frac{H(\mathbf{x}^*) - H(\widehat{\mathbf{x}^*})}{H(\mathbf{x}^*)}.$$

We run the simulation for 100 trials and compute the average error for each case. The Figure 3(A) plots the error for the grid of size $10 \times 10$, while Figure 3(B) plots the error for the grid of size $100 \times 10$.

Remind that the approximation guarantee of Theorem 1 is an error bound for the worst case. As the simulation result suggests, for any graph and any range of $\alpha$, the error of the local update algorithm decreases dramatically as $r$ increases. Moreover, when $r$ is comparably small as $r = 3$, the output of the local update algorithm achieves remarkably good approximation. Hence we observe that our algorithm performs well not only theoretically, but also practically.

**Setup 2.** We consider the vertex weighted independent set model defined on a grid graph. To this end, we start by description of a weighted independent set problem as the MRF model. Specifically, consider a binary MRF on an $n_1 \times n_2$ grid $G = (V, E)$:

$$\Pr(\mathbf{x}) \propto \exp\left(\sum_{i \in V} \theta_i x_i + \sum_{(i,j) \in E} \Psi(x_i x_j)\right), \quad \text{for } \mathbf{x} \in \{0,1\}^{n_1 n_2}.$$

Here, the parameters are chosen as follows.

1. For each $i \in V$, $\theta_i$ is chosen independently as per the distribution $\mathcal{U}[0,1]$.
2. The function $\Psi(\cdot, \cdot)$ is defined as

$$\Psi(\sigma, \sigma') = \begin{cases} -M & \text{if } (\sigma, \sigma') = (1,1) \\ 0 & \text{otherwise} \end{cases},$$

where $M$ is a large number.

For this model, we did simulations for grid graphs of size $10 \times 10$, $30 \times 10$, and $100 \times 10$ respectively. For each graph, we computed the average error as in the Setup 1, over 100 trials. The result is shown in the following table. As the result shows, our local update algorithm achieves remarkably good approximation of the MAP or equivalently in this setup the maximum weight independent set, even with very small $r$ values !

|  | $10 \times 10$ | $30 \times 10$ | $100 \times 10$ |
|---|---|---|---|
| r=1 | 0.219734 | 0.205429 | 0.208446 |
| r=2 | 0.016032 | 0.019145 | 0.019305 |
| r=3 | 0.001539 | 0.002616 | 0.002445 |

It is worth nothing that choosing $\theta_i$ from $\mathcal{U}[0, \alpha]$ for any $\alpha > 0$ will give the same approximation result, since $\mathbf{x}^*$ and $\widehat{\mathbf{x}}$ are both linear on $\alpha$.

## 6  Conclusion

We considered the question of designing simple, iterative algorithm with local updates for finding MAP in any pair-wise MRF. As the main result of this paper, we presented such a randomized, local iterative algorithm that can find $\varepsilon$-approximate solution of MAP in any pair-wise MRF based on $G$ within $2n \ln n$ iterations and the computation per iteration is constant $C(\varepsilon, \rho)$ dependent on the accuracy parameter $\varepsilon$ as well as the growth rate $\rho$ of the polynomially growing graph $G$. That is, ours is a local, iterative randomized PTAS for MAP problem in MRF with geometry. Our results are somewhat surprising given that thus far the known theoretical justification for such local algorithms strongly dependended on some form of convexity of the 'energy' function. In contrast, our results *do not* require any such condition, but only the geometry of the underlying MRF. We believe that our algorithm will be of great practical interest in near future as a large class of problems that utilize MRF based modeling and inference in practice have the underlying graphical structure possessing some form of geometry naturally.

## Footnotes

[1] We assume the positivity of $\Psi_v$'s and $\Psi_{uv}$'s for simplicity of analysis.

[2]Though this setup has $\phi_i, \psi_{ij}$ taking negative values, they are equivalent to the setup considered in the paper, since *affine* shift will make them non-negative without changing the distribution.

# References

[1] M. Bayati, D. Shah, and M. Sharma. Maximum weight matching via max-product belief propagation. In *IEEE ISIT*, 2005.

[2] M. Bayati, D. Shah, and M. Sharma. Max-Product for Maximum Weight Matching: Convergence, Correctness, and LP Duality. *IEEE Transactions on Information Theory*, 54(3):1241–1251, 2008.

[3] Y. Boykov, O. Veksler, and R. Zabih. Fast approximate energy minimization via graph cuts. *IEEE Trans. Pattern Anal. Mach. Intell.*, 23(11):1222–1239, 2001.

[4] William Feller. *An Introduction to Probability Theory and Its Applications*. Wiley, 1957.

[5] Hans-Otto Georgii. *Gibbs measures and phase transitions*. Walter de Gruyter, 1988.

[6] A. Gupta, R. Krauthgamer, and J.R. Lee. Bounded geometries, fractals, and low-distortion embeddings. In *In Proceedings of the 44th annual Symposium on the Foundations of Computer Science*, 2003.

[7] S. Har-Peled and M. Mendel. Fast construction of nets in low dimensional metrics, and their applications. In *Proceedings of the twenty-first annual symposium on Computational geometry*, pages 150–158. ACM New York, NY, USA, 2005.

[8] B. Huang and T. Jebara. Loopy belief propagation for bipartite maximum weight b-matching. *Artificial Intelligence and Statistics (AISTATS)*, 2007.

[9] N. Komodakis and G. Tziritas. A new framework for approximate labeling via graph cuts. In *International Conference on Computer Vision*, pages 1018–1025, 2005.

[10] M. Pawan Kumar and Philip H. S. Torr. Improved moves for truncated convex models. In *NIPS*, pages 889–896, 2008.

[11] Stan Z. Li. *Markov Random Field Modeling in Image Analysis*. Springer, 2001.

[12] M. Malfait and D. Roose. Wavelet-based image denoising using a markov random field a priori model. *IEEE Transactions on : Image Processing*, 6(4):549–565, 1997.

[13] Christopher D. Manning and Hinrich Schutze. *Foundations of Statistical Natural Language Processing*. The MIT Press, 1999.

[14] J. Pearl. *Probabilistic Reasoning in Intelligent Systems: Networks of Plausible Inference*. San Francisco, CA: Morgan Kaufmann, 1988.

[15] Thomas Richardson and Ruediger Ubanke. *Modern Coding Theory*. Cambridge University Press, 2008.

[16] S. Sanghavi, D. Shah, and A. Willsky. Message-passing for Maximum Weight Independent Set. In *Proceedings of NIPS*, 2007.

[17] R. Swendsen and J. Wang. Nonuniversal critical dynamics in monte carlo simulations. *Phys. Rev. Letter.*, 58:86–88, 1987.

[18] O. Veksler. Graph cut based optimization for mrfs with truncated convex priors. In *CVPR*, 2007.

[19] Paul Viola and Michael J. Jones. Robust real-time face detection. *International Journal of Computer Vision*, 57(2):137–154, 2004.

[20] M. Wainwright and M. Jordan. Graphical models, exponential families, and variational inference. *UC Berkeley, Dept. of Statistics, Technical Report 649*, 2003.

[21] M. J. Wainwright, T. Jaakkola, and A. S. Willsky. Map estimation via agreement on (hyper)trees: Message-passing and linear-programming approaches. *IEEE Transactions on Information Theory*, 2005.

[22] J. Yedidia, W. Freeman, and Y. Weiss. Generalized belief propagation. *Mitsubishi Elect. Res. Lab., TR-2000-26*, 2000.

